# Sparse Coding for Learning Interpretable Spatio-Temporal Primitives

**Taehwan Kim**
TTI Chicago
taehwan@ttic.edu

**Gregory Shakhnarovich**
TTI Chicago
gregory@ttic.edu

**Raquel Urtasun**
TTI Chicago
rurtasun@ttic.edu

## Abstract

Sparse coding has recently become a popular approach in computer vision to learn dictionaries of natural images. In this paper we extend the sparse coding framework to learn interpretable spatio-temporal primitives. We formulated the problem as a tensor factorization problem with tensor group norm constraints over the primitives, diagonal constraints on the activations that provide interpretability as well as smoothness constraints that are inherent to human motion. We demonstrate the effectiveness of our approach to learn interpretable representations of human motion from motion capture data, and show that our approach outperforms recently developed matching pursuit and sparse coding algorithms.

## 1 Introduction

In recent years sparse coding has become a popular paradigm to learn dictionaries of natural images [10, 1, 4]. The learned representations have proven very effective in computer vision tasks such as image denoising [4], inpainting [10, 8] and object recognition [1]. In these approaches, sparse coding was formulated as the sum of a data fitting term, typically the Frobenius norm, and a regularization term that imposes sparsity. The $\ell_1$ norm is typically used as it is convex instead of other sparsity penalties such as the $\ell_0$ pseudo-norm.

However, the sparsity induced by these norms is local; The estimated representations are sparse in that most of the activations are zero, but the sparsity has no structure, i.e., there is no preference to which coefficients are active. Mairal et al. [9] extend the sparse coding formulation of natural images to impose structure by first clustering the set of image patches and then learning a dictionary where members of the same cluster are encouraged to share sparsity patterns. In particular, they use group norms so that the sparsity patterns are shared within a group.

Here we are interested in the problem of learning dictionaries of human motion. Learning spatio-temporal representations of motion has been addressed in the neuroscience and motor control literature, in the context of motor synergies [13, 5, 14]. However, most approaches have focussed on learning static primitives, such as those obtained by linear subspace models applied to individual frames of motion [12, 15].

One notable exception to this is the work of diAvella et al. [3] where the goal was to recover primitives from time series of EMG signals recorded from a set of frog muscles. Using matching pursuit [11] and an $\ell_0$-type regularization as the underlying mechanism to learn primitives, [3] performed matrix factorization of the time series. The recovered factors represent the primitive dictionary and the primitive activations. However, this technique suffers from the inherent limitations of the $\ell_0$ regularization which is combinatorial in nature and thus difficult to optimize; therefore [3] resorted to a greedy algorithm that is subject to the inherent limitations of such an approach.

In this paper we propose to extend the sparse coding framework to learn motion dictionaries. In particular, we cast the problem of learning spatio-temporal primitives as a tensor factorization prob-

lem and introduce tensor group norms over the primitives that encourage sparsity in order to learn the number of elements in the dictionary. The introduction of additional diagonal constraints in the activations, as well as smoothness constraints that are inherent to human motion, will allow us to learn interpretable representations of human motion from motion capture data. As demonstrated in our experiments, our approach outperforms state-of-the-art matching pursuit [3], as well as recently developed sparse coding algorithms [7].

## 2 Sparse coding for motion dictionary learning

In this section we first review the framework of sparse coding, and then show how to extend this framework to learn interpretable dictionaries of human motion.

### 2.1 Traditional sparse coding

Let $\mathbf{Y} = [\mathbf{y}_1, \cdots, \mathbf{y}_N]$ be the matrix formed by concatenating the set of training examples drawn i.i.d. from $p(\mathbf{y})$. Sparse coding is usually formulated as a matrix factorization problem composed of a data fitting term, typically the Frobenius norm, and a regularizer that encourages sparsity of the activations

$$\min_{\mathbf{W},\mathbf{H}} ||\mathbf{Y} - \mathbf{WH}||_F^2 + \lambda \psi(\mathbf{H}) \ .$$

or equivalently

$$\min_{\mathbf{W},\mathbf{H}} \quad ||\mathbf{Y} - \mathbf{WH}||_F^2$$
$$\text{subject to} \quad \psi(\mathbf{H}) \leq \delta_{sparse}$$

where $\lambda$ and $\delta_{sparse}$ are parameters of the model. Additional bounding constraints on $\mathbf{W}$ are typically employed since there is an ambiguity on the scaling of $\mathbf{W}$ and $\mathbf{H}$. In this formulation $\mathbf{W}$ is the dictionary, with $\mathbf{w}_i$ the dictionary elements, $\mathbf{H}$ is the matrix of activations, and $\psi(\mathbf{H})$ is a regularizer that induces sparsity. Solving this problem involves a non-convex optimization. However, solving with respect to $\mathbf{W}$ and $\mathbf{H}$ alone is convex if $\psi$ is a convex function of $\mathbf{H}$. As a consequence, $\psi$ is usually taken to be the $\ell_1$ norm, i.e., $\psi(\mathbf{H}) = \sum_{i,j} |h_{i,j}|$, and an alternate minimization scheme is typically employed [7].

If the problem has more structure, one would like to use this structure in order to learn non-local sparsity patterns. Mairal et al. [9] exploit group norm sparsity priors to learn dictionaries of natural images by first clustering the training image patches, and then learning a dictionary where members of the same cluster are encouraged to share sparsity patterns. In particular, they use the $\ell_{2,1}$ norm defined as $\psi(\mathbf{H}) = \sum_k ||\mathbf{h}^k||_2$, where $\mathbf{h}^k$ are the elements of $\mathbf{H}$ that are members of the $k$-th group. Note that the members of a group do not need to be rows or columns, more complex group structures can be employed [6].

However, the structure imposed by these group norms is not sufficient for learning interpretable motion primitives. We now show how in the case of motion, we can consider the activations and the primitives as tensors and impose group norm sparsity on the tensors. Moreover, we impose additional constraints such as continuity and differentiability that are inherent of human motion data, as well as diagonal constraints that ensure interpretability.

### 2.2 Motion dictionary learning

Let $\mathbf{Y} \in \Re^{D \times L}$ be a D dimensional signal of temporal length $L$. We formulate the problem of learning dictionaries of human motion as a tensor factorization problem where the matrix $\mathbf{W}$ is now a tensor, $\mathbf{W} \in \Re^{D \times P \times Q}$, encoding temporal and spatial information, with $D$ the dimensionality of the observations, $P$ the number of primitives, and $Q$ the length of the primitives. $\mathbf{H}$ is now also defined as a tensor, $\mathbf{H} \in \Re^{Q \times P \times L}$, with $L$ the temporal length of the sequence. For simplicity in the discussion we assume that the primitives have the same length. This restriction can be easily removed by setting $Q$ to be the maximum length of the primitives and padding the remaining elements to zero. We thus define the data term to be

$$\ell_{data} = ||\mathbf{Y} - vec(\mathbf{W})vec(\mathbf{H})||_F \tag{2}$$

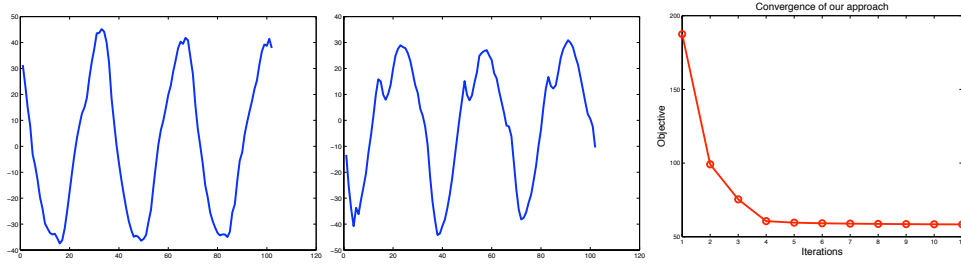

Figure 1: Walking dataset composed of multiple walking cycles performed by the same subject. (left, center) Projection of the data onto the first two principal components of walking. This is the data to be recovered. (right) Training error as a function of the number of iterations. Note that our approach converges after only a few iterations

where $vec(\mathbf{W}) \in \Re^{D \times PQ}$ and $vec(\mathbf{H}) \in \Re^{QP \times L}$ are projections of the tensors to be represented as matrices, i.e., flattening.

When learning dictionaries of human motion, there is additional structure and constraints that one would like the dictionary elements to satisfy. One important property of human motion is that it is smooth. We impose continuity and differentiability constraints by adding a regularization term that encourages smooth curvature, i.e., $\phi(\mathbf{W}) = \sum_{p=1}^{P} ||\nabla^2 \mathbf{W}_{p,:,:}||_F$.

One of the main difficulties with learning motion dictionaries is that the dictionary words might have very different temporal lengths. Note that this problem does not arise in traditional dictionary learning of natural images, since the size of the dictionary words is manually specified [4, 1, 9]. This makes the learning problem more complex since one would like to identify not only the number of elements in the dictionary, but also the size of each dictionary word. We address this problem by adding a regularization term that prefers dictionaries with small number of primitives, as well as primitives of short length. In particular, we extend the group norms over matrices to be group norms over tensors and define

$$\ell_{p,q,r}(\mathbf{W}) = \left( \sum_{i=1}^{P} \left( \sum_{j=1}^{Q} \left( \sum_{k=1}^{D} |W_{i,j,k}|^p \right)^{q/p} \right)^{r/q} \right)^{1/r}$$

where $W_{i,j,k}$ is the $k$-th dimension at the $j$-th time frame of the $i$-th primitive in $\mathbf{W}$.

We will also like to impose additional constraints on the activations $\mathbf{H}$. For interpretability, we would like to have only positive activations. Moreover, since the problem is under-constrained, i.e., $\mathbf{H}$ and $\mathbf{W}$ can be recovered up to an invertible transformation $\mathbf{WH} = (\mathbf{WC}^{-1})(\mathbf{CH})$, we impose that the elements of the activation tensor should be in the unit interval, i.e., $\mathbf{H}_{i,j,k} \in [0, 1]$. As in traditional sparse coding, we encourage the activations to be sparse. We impose this by bounding the $L_1$ norm. Finally, to impose interpretability of the results as spatio-temporal primitives, we impose that when a spatio-temporal primitive is active, it should be active across all its time-length with constant activation strength, i.e., $\forall i, j, k, \quad H_{i,j,k} = H_{i,j+1,k+1}$.

We thus formulate the problem of learning motion dictionaries as the one of solving the following optimization problem

$$\min_{\mathbf{W},\mathbf{H}} \quad ||\mathbf{Y} - vec(\mathbf{W})vec(\mathbf{H})||_F + \lambda\phi(\mathbf{W}) + \eta L_{p,q,r}(\mathbf{W})$$

subject to $\quad \forall i, j, k \quad 0 \leq H_{i,j,k} \leq 1, \quad H_{i,j,k} = H_{i,j+1,k+1}, \quad \forall j \quad \sum_{i,j} H_{i,j,k} \leq \delta_{train}$ (3a)

where $\delta_{train}$, $\lambda$ and $\eta$ are parameters of our model.

When optimizing over $\mathbf{W}$ or $\mathbf{H}$ alone the problem is convex. We thus perform alternate minimizatio. Our algorithm converges to a local minimum, the proof is similar to the convergence proof of block coordinate descent, see Prop. 2.7.1 in [2].

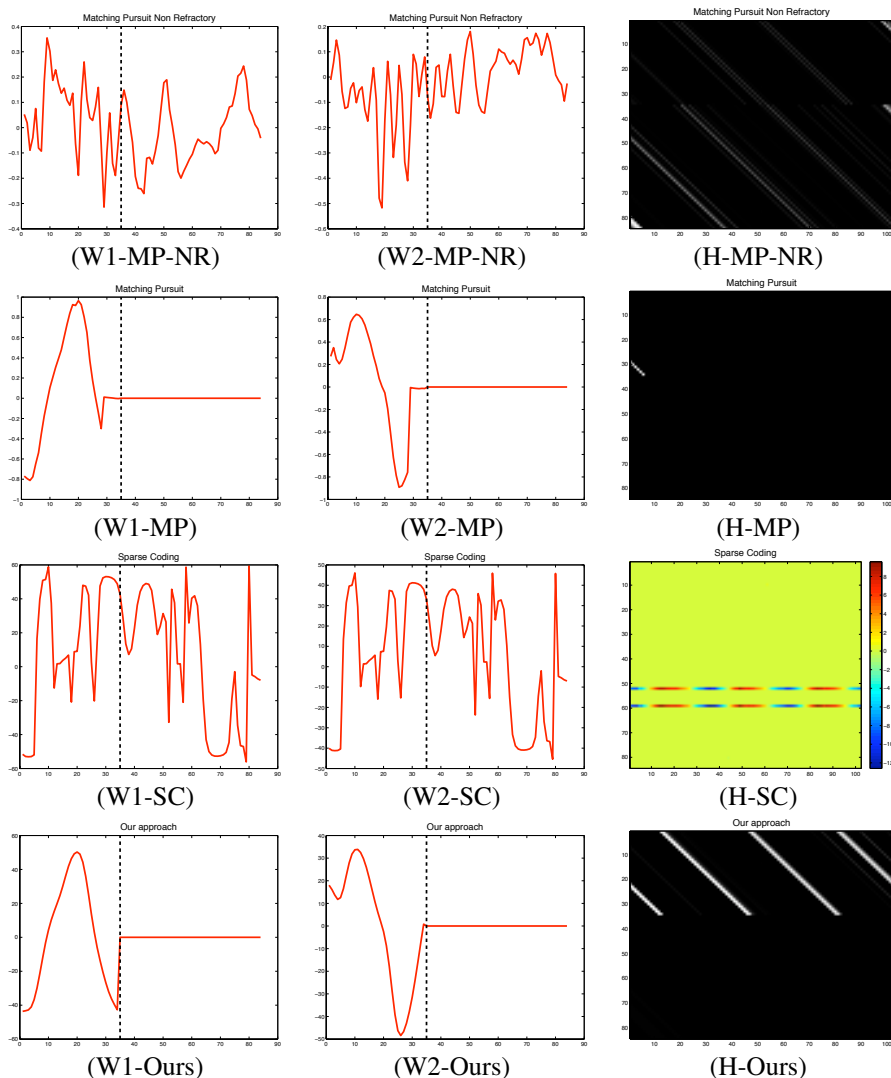

Figure 2: Estimation of **W** and **H** when the number of primitives is unknown, using (top) matching pursuit without refractory period, (second row) matching pursuit with refractory period [3], (third row) traditional sparse coding and (bottom) our approach. Note that our approach is able to recover the primitives, their number and the correct activations. Matching pursuit is able to recover the number of primitives when using refractory period, however the activations and the primitives are not correct. When we do not use the refractory period, the recovered primitives are very noisy. Sparse coding has a low reconstruction error, but neither the number of primitives, nor the primitives and the activations are correctly recovered.

## 3   Experimental Evaluation

We compare our algorithm to two state-of-the-art approaches in the task of discovering interpretable primitives from motion capture data, namely, the sparse coding approach of [7] and matching pursuit [3]. In the following, we first describe the baselines in detail. We then demonstrate our method's ability to estimate the primitives, their number, as well as the activation patterns. We then show that our approach outperforms matching pursuit and sparse coding when learning dictionaries of walking and running motions. For all experiments we set $\delta_{train} = 1$, $\delta_{test} = 1.3$, $\lambda = 1$ and $\eta = 0.05$ and use the $\ell_{2,1,1}$ norm. Note that similar results where obtained with the $\ell_{2,2,1}$ norm. For SC we use $\beta = 0.01$ and $c$ is set to the maximum value of the $\ell_2$ norm. The threshold for MP with refractory period is set to $0.1$.

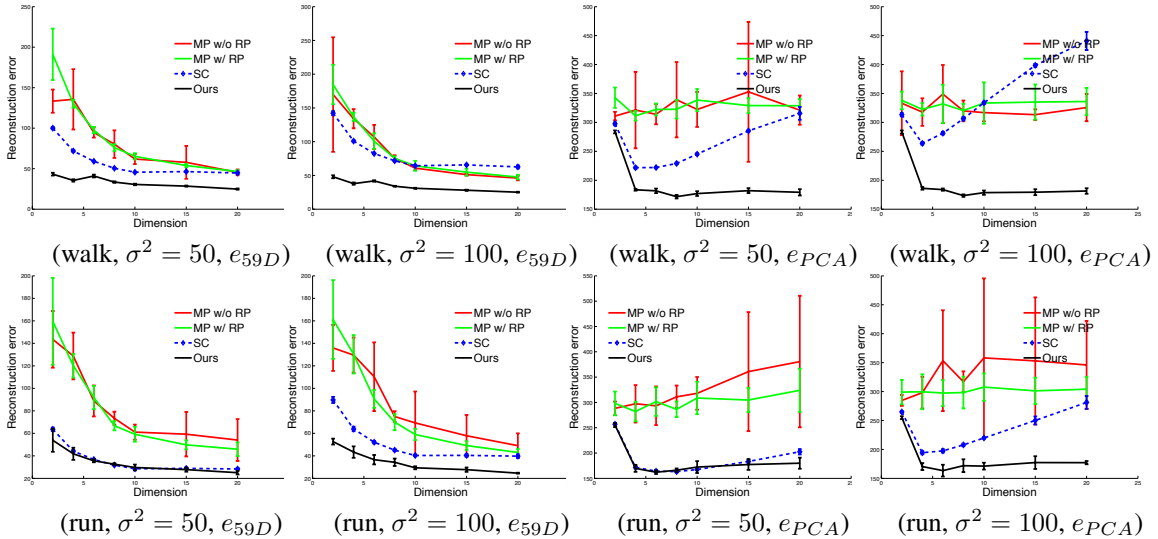

(walk, $\sigma^2 = 50$, $e_{59D}$)  (walk, $\sigma^2 = 100$, $e_{59D}$)  (walk, $\sigma^2 = 50$, $e_{PCA}$)  (walk, $\sigma^2 = 100$, $e_{PCA}$)

(run, $\sigma^2 = 50$, $e_{59D}$)  (run, $\sigma^2 = 100$, $e_{59D}$)  (run, $\sigma^2 = 50$, $e_{PCA}$)  (run, $\sigma^2 = 100$, $e_{PCA}$)

Figure 3: Error as a function of the dimension when adding Gaussian noise of variance 50 and 100. (Top) Walking, (bottom) running.

**Matching pursuit (MP):** We follow a similar approach to [3] where an alternate minimization over $\mathbf{W}$ and $\mathbf{H}$ is employed. For each iteration in the alternate minimization, $\mathbf{W}$ is optimized by minimizing $\ell_{data}$ defined in Eq. (2) until convergence. For each iteration in the optimization of $\mathbf{H}$, an over-complete dictionary $\mathcal{D}$ is created by taking the primitives in $\mathbf{W}$, and generating candidates by shifting each primitive in time. Note that the cardinality of the candidate dictionary is $|\mathcal{D}| = P(L + Q - 1)$ if $\mathbf{W}$ has $P$ primitives and the data is composed of $L$ frames. Once the dictionary is created, a set of primitives is iteratively selected (one at a time) by choosing at each iteration the primitive with the largest scalar product with respect to the residual signal that cannot be explained with the already selected primitives. Primitives are chosen until a threshold on the scalar product is reached. Note that this is an instance of Matching Pursuit [11], a greedy algorithm to solve an $\ell_0$-type optimization. Additionally, in the step of choosing elements in the dictionary, [3] introduced the refractory period, which means that when one element in the dictionary is chosen, all overlapping elements are removed from the dictionary. This is done to avoid multiple activations of primitives. In our experiments we compare our approach to matching pursuit with and without refractory period.

**Sparse coding (SC):** We use the sparse coding formulation of [7] which minimizes the Frobenius norm with an $L1$ regularization penalty on the activations

$$\min_{\bar{\mathbf{W}},\bar{\mathbf{H}}} \quad ||\mathbf{Y} - \bar{\mathbf{W}}\bar{\mathbf{H}}||_F \ + \ \beta \sum_{i,j} |\bar{H}_{i,j}|$$

$$\text{subject to} \quad \forall j \quad |\bar{\mathbf{W}}_{:,j}| \le c$$

with $\beta$ a constant trading off the relative influence of the data fitting term and the regularizer, and $c$ a constant bounding the value of the primitives. Note that now $\bar{\mathbf{W}}$ and $\bar{\mathbf{H}}$ are matrices. Following [7], we solve this optimization problem alternating between solving with respect to the primitives $\bar{\mathbf{W}}$ and the activations $\bar{\mathbf{H}}$.

### 3.1 Estimating the number of primitives

In the first experiment we demonstrate the ability of our approach to infer the number of primitives as well as the length of the existing primitives. For this purpose we created a simple dataset which is composed of a single sequence of multiple walking cycles performed by the same subject from the CMU mocap dataset [1]. We apply PCA to the data reducing the dimensionality of the observations

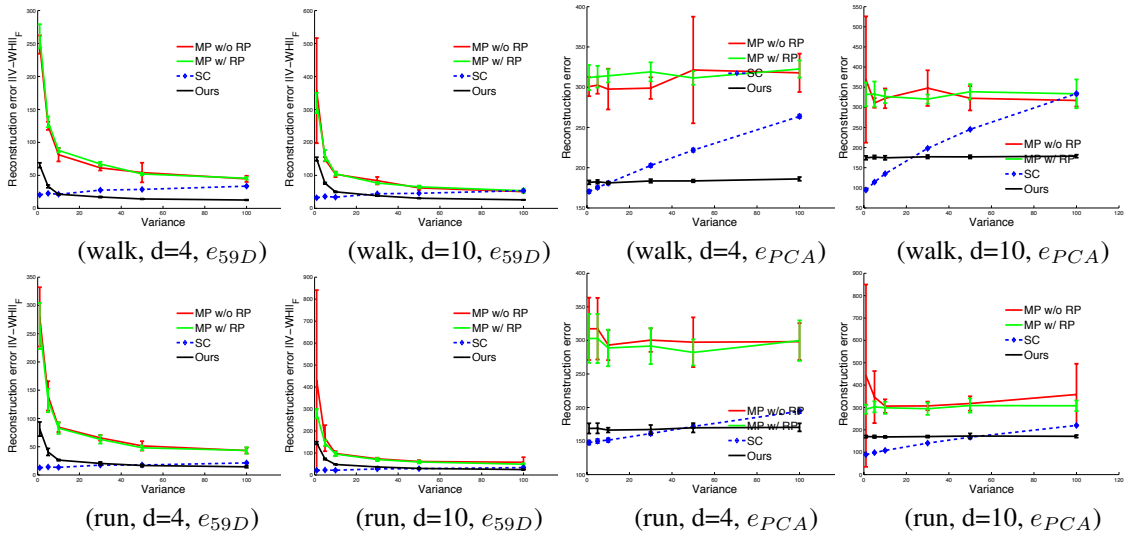

Figure 4: Error as a function of the Gaussian noise variance for 4D and 10D spaces learned from a dataset composed of a single subject. (Top) walking, (bottom) running.

from 59D to 2D for each time instant. Fig. 1 depicts the projections of the data onto the first two principal components as a function of time. In this case it is easy to see that since the motion is periodic, the signal could be represented by a single 2D primitive whose length is equal to the length of the period.

To perform the experiments we initialize our approach and the baselines with a sum of random smooth functions (sinusoids) whose frequencies are different from the principal frequency of the periodic training data, and set the number of primitives to $P = 2$. One primitive is set to have approximately the same length as a cycle of the periodic motion and the other primitive is set to be 50% larger. Note that a rough estimate of the length of the primitives could be easily obtained by analyzing the principal frequencies of the signal. Fig. 2 depicts the results obtained by our approach and the baselines. The first two columns depict the two dimensional primitives recovered (W1 and W2). Each plot represents $vec(\mathbf{W}_{i,:,:}) \in \Re^{(Q_1+Q_2)\times 1}$. The dotted black line separates the two primitives. Note that we expect these primitives to be similar to the original signal, i.e., $vec(\mathbf{W}_{1,:,:})$ similar to a period in Fig. 1 (left) and $vec(\mathbf{W}_{2,:,:})$ to a period in Fig. 1 (right). The third column depicts the activations $vec(\mathbf{H}) \in \Re^{(Q_1+Q_2)\times L}$ recovered. We expect the successful activations to be diagonal, and to appear only once every cycle.

Note that our approach is able to recover the number of primitives as well as the primitive themselves and the correct activations. Matching pursuit without refractory period (first row) is not able to recover the primitives, their number, or the activations. Moreover, the estimated signal has high frequencies. Matching pursuit with refractory period (second row) is able to recover the number of primitives, however the activations are underestimated and the primitives are not very accurate. Sparse coding has a low reconstruction error, but neither the primitives, their number, nor the activations are correctly recovered. This confirms the inability of traditional sparse coding to recover interpretable primitives, and the importance of having interpretability constraints such as the refractory period of matching pursuit and our diagonal constraints. Note also that as shown in Fig. 1 (right) our approach converges in a few iterations.

## 3.2 Quantitative analysis and comparisons

We evaluate the capabilities of our approach to reconstruct new sequences, and compare our approach to the baselines [3, 7] in a denoising scenario as well as when dealing with missing data. We preprocess the data by applying PCA to reduce the dimensionality of the input space. We measure error by computing the Frobenius norm between the test sequences and the reconstruction given by

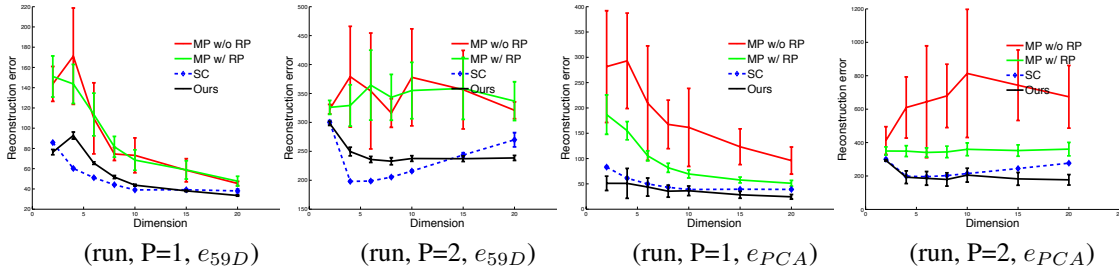

| (run, P=1, $e_{59D}$) | (run, P=2, $e_{59D}$) | (run, P=1, $e_{PCA}$) | (run, P=2, $e_{PCA}$) |

Figure 5: Multiple subject error as a function of the dimension for noisy data with variance 100 and different numbers of primitives. As expected one primitive is not enough for accurate reconstruction.

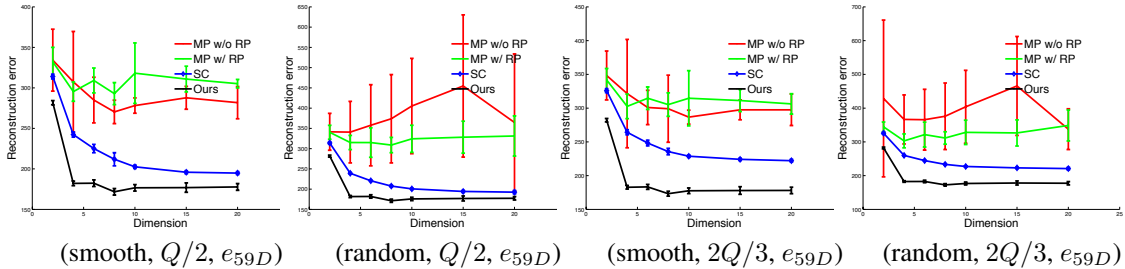

| (smooth, $Q/2$, $e_{59D}$) | (random, $Q/2$, $e_{59D}$) | (smooth, $2Q/3$, $e_{59D}$) | (random, $2Q/3$, $e_{59D}$) |

Figure 6: Missing data and influence of initialization: Error in the $59D$ space when $Q/2$ and $2Q/3$ of the data is missing. The primitives are either initialize randomly or to a smooth set of sinusoids of random frequencies.

the learned $\mathbf{W}$ and the estimated activations $\mathbf{H}_{test}$

$$e_{pca} = \frac{1}{D}||V_{test} - vec(\mathbf{W})vec(\mathbf{H}_{test})||_F$$

as well as the error in the original 59D space which can be computed by projecting back into the original space using the singular vectors. Note that $\mathbf{W}$ is learned at training, and the activations $\mathbf{H}_{test}$ are estimated at inference time. To evaluate the generalization properties of each algorithm, we compute both errors in a denoising scenario, where $\mathbf{H}_{test}$ is obtained using $\hat{\mathbf{V}}_{test} = \mathbf{V}_{test} + \epsilon$, with $\epsilon$ i.i.d Gaussian noise, and the errors are computed using the ground truth data $\mathbf{V}_{test}$. For each experiment we use $P = 1$, $\eta = 0.05$, $\delta_{train} = 1$, $\delta_{test} = 1.3$ and a rough estimate of $Q$, which can be easily obtained by examining the principal frequencies of the data [16]. The primitives are initialized to a sum of sinusoids of random frequencies.

We created a walking dataset composed of motions performed by the same subject. In particular we used motions $\{02, 03, 04, 05, 06, 07, 08, 09, 10, 11\}$ of subject 35 in the CMU mocap dataset. We also performed reconstruction experiments for running motions and used motions $\{17, 18, 20, 21, 22, 23, 24, 25\}$ from subject 35. In both cases, we use 2 sequences for training and the rest for testing, and report average results over 10 random splits. Fig. 3 depicts reconstruction error in PCA space and in the original space as a function of the noise variance. Fig. 4 depicts reconstruction error as a function of the dimensionality of the PCA space. Our approach outperforms matching pursuit with and without refractory period in all scenarios. Note that out method outperforms sparse coding when the output is noisy. This is due to the fact that, given a big enough dictionary, sparse coding overfits and can perfectly fit the noise.

We also performed reconstruction experiments for running motions performed by different subjects. In particular we use motions $\{03, 04, 05, 06\}$ of subject 9 and motions $\{21, 23, 24, 25\}$ of subject 35. Fig. 5 depicts reconstruction error for our approach when using different numbers of primitives. As expected one primitive is not enough for accurate reconstruction. When using two primitives our approach performs comparable to sparse coding and clearly outperforms the other baselines.

In the next experiment we show the importance of having interpretable primitives. In particular we compare our approach to the baselines in a missing data scenario, where part of the sequence is missing. In particular, $Q/2$ and $2Q/3$ frames are missing. We use the single subject walking database.

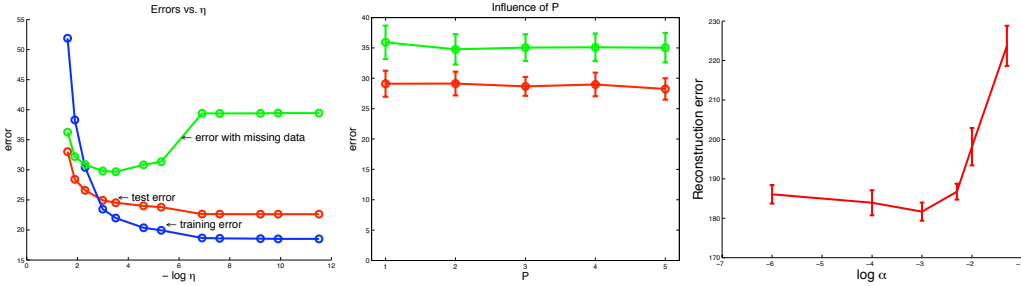

Figure 7: Influence of $\eta$ and $P$ on the single subject walking dataset as well as using soft constraints instead of hard constraints on the activations. (left) Our method is fairly insensitive to the choice of $\eta$. As expected the reconstruction error of the training data decreases when there is less regularization. The test error however is very flat, and increases when there is too much or too little regularization. For missing data, having good primitives is important, and thus regularization is necessary. Note that the horizontal axis depicts $-\log\eta$, thus $\eta$ decreases for larger values of this axis. (center) Error with (green) and without (red) missing data as a function of $P$. Our approach is not sensitive to the value of $P$; one primitive is enough for accurate reconstruction in this dataset. (right) Error when using solft constraints $|H_{i,j,k} - H_{i,j+1,k+1}| \leq \alpha$ as a function of $\alpha$. The leftmost point corresponds to $\alpha = 0$, i.e., $H_{i,j,k} = H_{i,j+1,k+1}$.

As shown in Fig. 6 our approach clearly outperforms all the baselines. This is due to the fact that sparse coding does not have structure, while the structure imposed by our equality constraints, i.e., $\forall i, j, k \ H_{i,j,k} = H_{i,j+1,k+1}$, help "hallucinate" the missing data. We also investigate the influence of initialization by using a random non-smooth initialization and the smooth initialization described above, i.e.,sinusoids of random frequencies. Note that as our approach, sparse coding is not sensitive to initialization. This is in contrast with MP which is very sensitive due to the $\ell_0$-type regularization.

We also investigated the influence of the amount of regularization on $\mathbf{W}$. Towards this end we use the single subject walking dataset, and compute reconstruction error for the training and test data with and without missing data as a function of $\eta$. As shown in Fig. 7 (left) our method is fairly insensitive to the choice of $\eta$. As expected the reconstruction error of the training data decreases when there is less regularization. The test error in the noiseless case is however very flat, and increases slightly when there is too much or too little regularization. When dealing with missing data, having good primitives becomes more important. Note that the horizontal axis depicts $-\log\eta$, thus $\eta$ decreases for larger values of the horizontal axis. The test error is higher than the training error for large $\eta$ since we use $\delta_{train} = 1$ and $\delta_{test} = 1.3$. Thus we are more conservative at learning since we want to learn interpretable primitives. We also investigate the sensitivity of our approach to the number of primitives. We use the single subject walking dataset and report errors averaged over 10 partitions of the data. As shown in Fig. 7 (middle) our approach is very insensitive to $P$; in this example a single primitive is enough for accurate reconstruction.

We finally investigate the influence of replacing the hard constraints on the activations by soft constraints $|H_{i,j,k} - H_{i,j+1,k+1}| \leq \alpha$. Note that our approach is not sensitive to the value of $\alpha$ and that the hard constraints ( $H_{i,j,k} = H_{i,j+1,k+1}$), depicted in the leftmost point in Fig. 7 (right), are almost optimal. This justifies our choice since when using hard constraints we do not need to search for the optimal value of $\alpha$.

## 4  Conclusion

We have proposed a sparse coding approach to learn interpretable spatio-temporal primitives of human motion. We have formulated the problem as a tensor factorization problem with tensor group norm constraints over the primitives, diagonal constraints on the activations, as well as smoothness constraints that are inherent to human motion. Our approach has proven superior to recently developed matching pursuit and sparse coding algorithms in the task of learning interpretable spatio-temporal primitives of human motion from motion capture data. In the future we plan to investigate applying similar techniques to learn spatio-temporal dictionaries of video data such as dynamic textures.

## Footnotes

[1]The data was obtained from mocap.cs.cmu.edu

# References

[1] S. Bengio, F Pereira, Y. Singer, and D. Strelow. Group sparse coding. In *NIPS*, 2009.

[2] D. P. Bertsekas. *Nonlinear Programming.* Athena Scientific, Belmont, Massachusetts, 1999.

[3] A. diAvella and E. Bizzi. Shared and specific muscle synergies in natural motor behaviors. *PNAS*, 102(8):3076–3081, 2005.

[4] M. Elad and M. Aharon. Image denoising via sparse and redundant representations over learned dictionaries. *IEEE Trans. on Image Processing*, 15(12):3736–3745, 2006.

[5] Z. Ghahramani. Building blocks of movement. *Nature*, 407:682–683, 2000.

[6] R. Jenatton, G. Obozinski, and F. Bach. Structured sparse principal component analysis. In *Proc. AIS-TATS10*, 2010.

[7] H. Lee, Alexis Battle, Raina R, and A. Y. Ng. Efficient sparse coding algorithms. In *NIPS*, 2007.

[8] J. Mairal, F. Bach, J. Ponce, and G. Sapiro. Online dictionary learning for sparse coding. In *ICML*, 2009.

[9] J. Mairal, F. Bach, J. Ponce, G. Sapiro, and A. Zisserman. Non-local sparse models for image restoration. In *ICCV*, 2009.

[10] J. Mairal, G. Sapiro, and M. Elad. Learning multiscale sparse representations for image and video restoration. *SIAM Multiscale Modelling and Simulation.*, 7(1):214–241, 2008b.

[11] S. G. Mallat and Z. Zhang. Matching pursuits with time-frequency dictionaries. *IEEE Trans. Signal. Proc. 41*, pages 3397–3415, 1993.

[12] C. R. Mason, J. E. Gomez, and T. J. Ebner. Hand synergies during reach to grasp. *J. of Neurophysiology*, 86:2896–2910, 2001.

[13] F. A. Mussa-Ivaldi and E. Bizzi. Motor learning: the combination of primitives. *Phil. Trans. Royal Society London, Series B*, 355:1755–1769, 2000.

[14] F. A. Mussa-Ivaldi and S. Solla. Neural primitives for motion control. *IEEE Journal on Ocean Engineering*, 29(3):640–650, 2004.

[15] E. Todorov and Z. Ghahramani. Analysis of the synergies underlying complex hand manipulation. In *Proceedings of Conf. of the IEEE Engineering in Medicine and Biology Society*, pages 4637–4640, 2004.

[16] R. Urtasun, D. J. Fleet, A. Geiger, J. Popovic, T. Darrell, and N. D. Lawrence. Topologically-constrained latent variable models. In *ICML*, 2008.

